# Training Factor Graphs with Reinforcement Learning for Efficient MAP Inference

**Michael Wick, Khashayar Rohanimanesh, Sameer Singh, Andrew McCallum**
Department of Computer Science
University of Massachusetts Amherst
Amherst, MA 01003
{*mwick,khash,sameer,mccallum*}*@cs.umass.edu*

## Abstract

Large, relational factor graphs with structure defined by first-order logic or other languages give rise to notoriously difficult inference problems. Because unrolling the structure necessary to represent distributions over all hypotheses has exponential blow-up, solutions are often derived from MCMC. However, because of limitations in the design and parameterization of the jump function, these sampling-based methods suffer from local minima—the system must transition through lower-scoring configurations before arriving at a better MAP solution. This paper presents a new method of explicitly selecting fruitful downward jumps by leveraging reinforcement learning (RL). Rather than setting parameters to maximize the likelihood of the training data, parameters of the factor graph are treated as a log-linear function approximator and learned with methods of temporal difference (TD); MAP inference is performed by executing the resulting policy on held out test data. Our method allows efficient gradient updates since only factors in the neighborhood of variables affected by an action need to be computed—we bypass the need to compute marginals entirely. Our method yields dramatic empirical success, producing new state-of-the-art results on a complex joint model of ontology alignment, with a 48% reduction in error over state-of-the-art in that domain.

## 1 Introduction

Factor graphs are a widely used representation for modeling complex dependencies amongst hidden variables in structured prediction problems. There are two common inference problems: learning (setting model parameters) and decoding (*maximum a posteriori* (MAP) inference). MAP inference is the problem of finding the most probable setting to the graph's hidden variables conditioned on some observed variables.

For certain types of graphs, such as chains and trees, exact inference and learning is polynomial time [1, 2, 3]. Unfortunately, many interesting problems require more complicated structure rendering exact inference intractable [4, 5, 6, 7]. In such cases we must rely on approximate techniques; in particular, stochastic methods such as Markov chain Monte Carlo (e.g., Metropolis-Hastings) have been applied to problems such as MAP inference in these graphs [8, 9, 10, 11, 6]. However, for many real-world structured prediction tasks, MCMC (and other local stochastic methods) are likely to struggle as they transition through lower-scoring regions of the configuration space.

For example, consider the structured prediction task of clustering where the MAP inference problem is to group data points into equivalence classes according to some model. Assume for a moment that

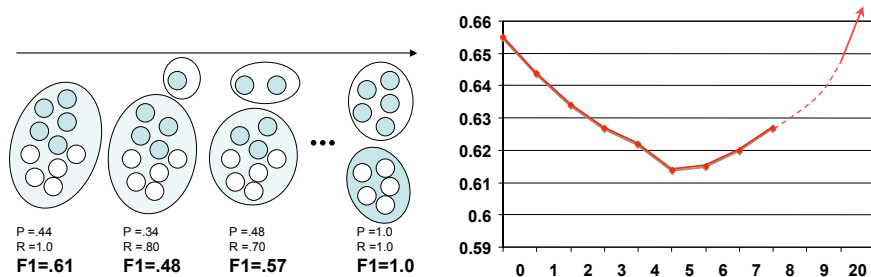

Figure 1: The figure on the left shows the sequence of states along an optimal path beginning at a single-cluster configuration and ending at the MAP configuration (F1 scores for each state are shown). The figure on the right plots the F1 scores along the optimal path to the goal for the case where the MAP clustering has forty instances (twenty per cluster) instead of 5.

this model is perfect and exactly reflects the pairwise F1 score. Even in these ideal conditions MCMC must make many downhill jumps to reach the MAP configuration. For example, Figure 1 shows the F1 scores of each state along the optimal path to the MAP clustering (assuming each MCMC jump can reposition one data point at a time). We can see that several consecutive downhill transitions must be realized before model-scores begin to improve.

Taking into account the above discussion with an emphasis on the delayed feedback nature of the MAP inference problem immediately inspires us to employ reinforcement learning (RL) [12]. RL is a framework for solving the sequential decision making problem with delayed reward. There has been an extensive study of this problem in many areas of machine learning, planning, and robotics. Our approach is to directly learn the parameters of the log-linear factor graph with reinforcement learning during a training phase; MAP inference is performed by executing the policy. Because we develop the reward-structure to assign the most mass to the goal configuration, the parameters of the model can also be interpreted as a regularized version of maximum likelihood that is smoothed over neighboring states in the proposal manifold.

The rest of this document is organized as follows: in §2 we briefly overview background material. In §3 we describe the details of our algorithm and discuss a number of ideas for coping with the combinatorial complexity in both state and action spaces. In §4.3 we present our empirical results, and finally in §6 we conclude and lay out a number of ideas for future work.

## 2 Preliminaries

### 2.1 Factor Graphs

A factor graph is undirected bipartite graphical representation of a probability distribution with random variables and factors as nodes. Let $X$ be a set of observed variables and $Y$ be a set of hidden variables. The factor graph expresses the conditional probability of $Y = \mathbf{y}$ given $X = \mathbf{x}$ discriminatively:

$$P(\mathbf{y}|\mathbf{x}) = \frac{1}{Z_X} \prod_{\psi_i \in \Psi} \psi_i(\mathbf{x}, y^i) = \frac{1}{Z_X} \exp\left(\sum_k \theta_k \phi_k(\mathbf{x}, y^k)\right) \tag{1}$$

Where $Z_X$ is an input-dependent normalizing constant ensuring that the distribution sums to one, $\Psi$ is the set of factors, and $\psi(x, y^i)$ are factors over the observed variables $\mathbf{x}$ and a set of hidden variables $y^i$ that are the neighbors of the factor (we use superscript to denote a set). Factors are log-linear combinations of features $\phi(\mathbf{x}, y^i)$ and parameters $\theta = \{\theta_j\}$. The problem of learning is to find a setting of the parameters $\theta$ that explains the data. For example, maximum likelihood sets the parameters so that the model's feature expectations matches the data's expectations.

## 2.2 Reinforcement Learning

Most of the discussion here is based on [12]. Reinforcement learning (RL) refers to a class of problems in which an agent interacts with the environment and the objective is to learn a course of actions that optimizes a long-term measure of a delayed reward signal. The most popular realization of RL has been in the context of markov decision processes (MDPs).

An MDP is the tuple $\mathcal{M} = \langle \mathcal{S}, \mathcal{A}, \mathcal{R}, \mathcal{P} \rangle$, where $\mathcal{S}$ is the set of states, $\mathcal{A}$ is the set of actions, $\mathcal{R} : \mathcal{S} \times \mathcal{A} \times \mathcal{S} \to \mathbb{R}$ is the reward function, i.e. $\mathcal{R}(s, a, s')$ is the expected reward when action $a$ is taken in state $s$ and transitions to state $s'$, and $\mathcal{P} : \mathcal{S} \times \mathcal{A} \times \mathcal{S} \to [0, 1]$ is the transition probability function, i.e. $\mathcal{P}^a(s, s')$ is the probability of reaching state $s'$ if action $a$ is taken in state $s$.

A stochastic policy $\pi$ is defined as $\pi : \mathcal{S} \times \mathcal{A} \to [0, 1]$ such that $\sum_a \pi(a|s) = 1$, where $\pi(s, a)$ is the probability of choosing action $a$ (as the next action) when in state $s$. Following a policy on an MDP results in an expected discounted reward $R_t^\pi$ accumulated over the course of the run, where $R_t^\pi = \sum_{k=0}^{T} \gamma^k r_{t+k+1}$. An optimal policy $\pi^\star$ is a policy that maximizes this reward.

Given a Q-function ($Q : \mathcal{S} \times \mathcal{A} \to \mathbb{R}$) that represents the expected discounted reward for taking action $a$ in state $s$, the optimal policy $\pi^\star$ can be found by locally maximizing $Q$ at each step. Methods of temporal difference (TD) [13] can be used to learn the optimal policy in MDPs, and even have convergence guarantees when the Q-function is in tabular form. However, in practice, tabular representations do not scale to large or continuous domains; a problem that function approximation techniques address [12]. Although the convergence properties of these approaches have not yet been established, the methods have been applied successfully to many problems [14, 15, 16, 17].

When linear functional approximation is used, the state-action pair $\langle s, a \rangle$ is represented by a feature vector $\phi(s, a)$ and the $Q$ value is represented using a vector of parameters $\theta$, i.e.

$$Q(s, a) = \sum_{\phi_k \in \phi(s,a)} \theta_k \phi_k \tag{2}$$

Instead of updating the $Q$ values directly, the updates are made to the parameters $\theta$:

$$\theta \quad \leftarrow \quad \theta + \alpha \left( r_{t+1} - Q(s_t, a_t) + \gamma \max_a Q(s_{t+1}, a) \right) \phi(s_t, a_t) \tag{3}$$

notice the similarity between the linear function approximator (Equation 2) and the log-linear factors (right-hand side of Equation 1); namely, the approximator has the same form as the unnormalized log probabilities of the distribution. This enables us to share the parameters $\theta$ from Equation 1.

## 3 Our Approach

In our RL treatment of learning factor graphs, each state in the system represents a complete assignment to the hidden variables $Y=y$. Given a particular state, an action modifies the setting to a subset of the hidden variables; therefore, an action can also be defined as a setting to all the hidden variables $Y=y'$. However, in order to cope with complexity of the action space, we introduce a *proposer* (as in Metropolis-Hastings) $\mathcal{B} : Y \to Y$ that constrains the space by limiting the number of possible actions from each state. The reward function $\mathcal{R}$ can be defined as the residual performance improvement when the systems transitions from a current state $y$ to a neighboring state $y'$ on the manifold induced by $\mathcal{B}$. In our approach, we use a performance measure based on the ground truth labels (for example, F1, accuracy, or normalized mutual information) as the reward. These rewards ensure that the ground truth configuration is the goal.

### 3.1 Model

Recall that an MDP is defined as $\mathcal{M} = \langle \mathcal{S}, \mathcal{A}, \mathcal{R}, \mathcal{P} \rangle$ with a set of states $\mathcal{S}$, set of actions $\mathcal{A}$, reward function $\mathcal{R}$ and transition probability function $\mathcal{P}$; we can now reformulate MAP inference and learning in factor graphs as follows:

• **States:** we require the state space to encompass the entire feasible region of the factor graph. Therefore, a natural definition for a state is a complete assignment to the hidden variables $Y \leftarrow y$ and

the state space itself is defined as the set $\mathcal{S} = \{y \mid y \in \mathrm{DOM}(Y)\}$, where $\mathrm{DOM}(Y)$ is the domain space of $Y$, and we omit the fixed observables $x$ for clarity since only $y$ is required to uniquely identify a state. Note that unless the hidden variables are highly constrained, the feasible regional will be combinatorial in $|Y|$; we discuss how to cope with this in the following sections.

- **Actions** Given a state $s$ (e.g., an assignment of $Y$ variables), an action may be defined as a constrained set of modifications to a subset of the hidden variable assignments. We constrain the action space to a manageable size by using a *proposer*, or a *behavior policy* from which actions are sampled. A proposer defines the set of reachable states by describing the distribution over neighboring states $s'$ given a state $s$. In context of the action space of an MDP, the proposer can be viewed in two ways. First, each possible neighbor state $s'$ can be considered the result of an action $a$, leading to a large number of deterministic actions. Second, it can be regarded as a single highly stochastic action, whose next state $s'$ is a sample from the distribution given by the proposer. Both of these views are equivalent; the former view is used for notation simplicity.

- **Reward Function** The reward function is designed so that the policy learned through delayed reward reaches the MAP configuration. Rewards are shaped to facilitate efficient learning in this combinatorial space. Let $\mathcal{F}$ be some performance metric (for example, for information extraction tasks, it could be $F1$ score based on the ground truth labels).

The reward function used is the residual improvement based on the performance metric $\mathcal{F}$ when the system transitions between states $s$ and $s'$:

$$R(s, s') = \mathcal{F}(s') - \mathcal{F}(s) \tag{4}$$

this reward can viewed as learning to minimize the geodesic distance between a current state and the MAP configuration on the proposal manifold. Alternatively, we could define a Euclidean reward as $\mathcal{F}(s^\star) - \mathcal{F}(s')$, where $s^\star$ is the ground truth. We choose an $F$ such that the ground truth scores the highest, that is $s^\star = \arg\max_s \mathcal{F}(s)$.

- **Transition Probability Function:** Recall that the actions in our system are samples generated from a proposer $\mathcal{B}$, and that each action uniquely identifies a next state in the system. The function that returns this next state deterministically is called *simulate*(s,a). Thus, given the state $s$ and the action $a$, the next state $s'$ has probability $\mathcal{P}^a(s, s') = 1$ if $s' = simulate(s, a)$, and 0 otherwise.

## 3.2 Efficient Q Value Computations

We use linear function approximation to obtain $Q$ values over the state/action space. That is, $Q(s, a) = \theta \cdot \phi(s, a)$, where $\phi(s, a)$ are features over the state-action pair $s, a$. We show below how $Q$ values can be derived from the factor graph (Equation 1) in a manner that enables efficient computations.

As mentioned previously, a state is an assignment to hidden variables $Y{=}y$ and an action is another assignment to the hidden variables $Y{=}y'$ (that results from changing the values of a subset of the variables $\Delta_Y \in Y$). Let $\delta_y$ be the setting to those variables in $y$ and $\delta_{y'}$ be the new setting to those variables in $y'$. For each assignment, the factor graph can compute the conditional probability $p(y \mid x)$. Then, the residual log-probability $S$ resulting from taking action $a$ in state $y$ and reaching $y'$ is therefore $\log(p(y' \mid x)) - \log(p(y \mid x))$. Plugging in the model from Equation 1 and performing some algebraic manipulation so redundant factors cancel yields:

$$\theta \cdot \left( \sum_{y'^i \in \delta_{y'}} \phi(x, y'^i) - \sum_{y^i \in \delta_y} \phi(x, y^i) \right) \tag{5}$$

Where the partition function $Z_X$ and factors outside the neighborhood of $\Delta_y$ cancel. In practice an action will modify a small subset of the variables so this computation is extremely efficient. We are now justified in using Equation 5 (derived from the model) to compute the inner product $(\theta \cdot \phi(s, a))$ from Equation 2.

### 3.3 Algorithm

Now that we have defined MAP inference in a factor graph as an MDP, we can apply a wide variety of RL algorithms to learn the model's parameters. In particular, we build upon Watkin's Q($\lambda$) [18, 19], a temporal difference learning algorithm [13]; we augment it with function approximation as described in the previous section. Our RL learning method for factor graphs is shown in Algorithm 1.

---

**Algorithm 1** Modified Watkin's-Q($\lambda$) for Factor Graphs

---

1: Input: Performance metric $\mathcal{F}$, proposer $\mathcal{B}$
2: Initialize $\overrightarrow{\theta}$ and $\overrightarrow{e} = \overrightarrow{0}$
3: **repeat** {For every episode}
4:     $s \leftarrow$ random initial configuration
5:     Sample $n$ actions $a \leftarrow \mathcal{B}(s)$; collect action samples in $A_{\mathcal{B}}(s)$
6:     **for** samples $a \in A_{\mathcal{B}}(s)$ **do**
7:       $s' \leftarrow simulate(s, a)$
8:       $\phi(s, s') \leftarrow$ set of features between $s, s'$
9:       $Q(s, a) \leftarrow \theta \cdot \phi(s, s')$     {Equation 5}
10:    **end for**
11:    **repeat** {For every step of the episode}
12:      **if** with probability $(1 - \epsilon)$ **then**
13:        $a \leftarrow \arg\max_{a'} Q(s, a')$
14:        $\overrightarrow{e} \leftarrow \gamma\lambda\overrightarrow{e}$
15:      **else**
16:        Sample a random action $a \leftarrow \mathcal{B}(s)$
17:        $\overrightarrow{e} \leftarrow \overrightarrow{0}$
18:      **end if**
19:      $s' \leftarrow simulate(s, a)$
20:      $\forall \phi_i \in \phi(s, s') : e(i) \leftarrow e(i) + \phi_i$    {Accumulate eligibility traces}
21:      Observe reward $r = \mathcal{F}(s) - \mathcal{F}(s')$    {Equation 4}
22:      $\delta \leftarrow r - Q(s, a)$
23:      Sample $n$ actions $a \leftarrow \mathcal{B}(s')$; collect action samples in $A_{\mathcal{B}}(s')$
24:      **for** samples $a \in A_{\mathcal{B}}(s')$ **do**
25:        $s'' \leftarrow simulate(s', a)$
26:        $\phi(s', s'') \leftarrow$ set of features between $s', s''$
27:        $Q(s', a) \leftarrow \theta \cdot \phi(s', s'')$
28:      **end for**
29:      $a \leftarrow \arg\max_{a'} Q(s', a')$
30:      $\delta \leftarrow \delta + \gamma Q(s', a)$    {Equation 3 with elig. traces}
31:      $\overrightarrow{\theta} \leftarrow \overrightarrow{\theta} + \alpha\delta\overrightarrow{e}$
32:      $s \leftarrow s'$
33:    **until** end of episode
34: **until** end of training

---

At the beginning of each episode, the factor graph is initialized to a random initial state $s$ (by assigning $Y = y_0$). Then, during each step of the episode, the maximum action is obtained by *repeatedly* sampling from the proposal distribution ($s' = simulate(s, a)$). The system transition to the greedy state $s'$ with high probability $(1 - \epsilon)$, or transitions to a random state instead. We also include eligibility traces that have been modified to handle function approximation [12].

Once learning has completed on a training set, MAP inference can be evaluated on test data by executing the resulting policy. Because $Q$-values encode both the reward and value together, policy execution can be performed by choosing the action that maximizes the $Q$-function at each state.

## 4 Experiments

We evaluate our approach by training a factor graph for solving the *ontology alignment* problem. Ontology alignment is the problem of mapping concepts from one ontology to semantically equivalent concepts from another ontology; our treatment of the problem involves learning a first-order probabilistic model that clusters concepts into semantically equivalent sets. For our experiments,

we use the the dataset provided by the Illinois Semantic Integration Archive (ISIA)[1]. There are two ontology mappings: one between two course catalog hierarchies, and another between two company profile hierarchies. Each ontology is organized as a taxonomy tree. The course catalog contains 104 concepts and 4360 data records while the company profile domain contains 219 concepts and 23139 records. For our experiments we perform two-fold cross validation with even splits.

The conditional random field we use to model the problem factors into binary decisions over sets of concepts, where the binary variable is *one* if all concepts in the set map to each other, and *zero* otherwise. Each of these hidden variables neighbors a factor that also examines the observed concept data. Since there are variables and factors for each hypothetical cluster, the size of the CRF is combinatorial in the number of concepts in the ontology, and it cannot be full instantiated even for small amounts of data. Therefore, we believe that this is be a good dataset demonstrate the scalability of the approach.

## 4.1 Features

The features used to represent the ontology alignment problem are described here. We choose to encode our features in first order logic, aggregating and quantifying pairwise comparisons of concepts over entire sets. These features are described more detail in our technical report [17].

The pairwise feature extractors are the following:
- TFIDF cosine similarity between concept-names of $c_i$ and $c_j$
- TFIDF cosine similarity between data-records that instantiate $c_i$ and $c_j$
- TFIDF similarity of the children of $c_i$ and $c_j$
- Lexical features for each string in the concept name
- True if there is a substring overlap between $c_i$ and $c_j$
- True if both concepts are the same level in the tree

The above pairwise features are used as a basis for features over entire sets with the following first order quantifiers and aggregators:

- $\forall$: universal first order logic quantifier
- $\exists$: existential quantifier
- Average: conditional mean over a cluster
- Max: maximum value obtained for a cluster
- Min: minimum value obtained for a cluster
- Bias: conditional bias, counts number of pairs where a pairwise feature could potentially fire.

The real-valued aggregators (min,max,average) are also quantized into bins of various sizes corresponding to the number of bins=$\{2,4,20,100\}$. Note that our first order features must be computed on-the-fly since the model is too large to be grounded in advance.

|        | Course Catalog | | | Company Profile | | |
|--------|------|-----------|--------|------|-----------|--------|
|        | F1   | Precision | Recall | F1   | Precision | Recall |
| RL     | **94.3** | 96.1  | 92.6   | **84.5** | 84.5  | 84.5   |
| MH-CD1 | 76.9 | 78.0      | 57.0   | 64.7 | 64.7      | 64.7   |
| MH-SR  | 92.0 | 88.9      | 76.3   | 81.5 | 88.0      | 75.9   |
| GA-PW  | 89.9 | 100       | 81.5   | 81.5 | 88.0      | 75.9   |
| GLUE   | 80   | 80        | 80     | 80   | 80        | 80     |

Table 1: pairwise-matching precision, recall and F1 on the course catalog dataset

## 4.2 Systems

In this section we evaluate the performance of our reinforcement learning approach to MAP inference and compare it current stochastic and greedy alternatives. In particular, we compare piecewise [20], contrastive divergence [21], and SampleRank [22, 11, 23]; these are described in more detail below.

- **Piecewise (GA-PW):** the CRF parameters are learned by training independent logistic regression classifiers in a piecewise fashion. Inference is performed by greedy agglomerative clustering.

- **Contrastive Divergence (MH-CD1) with Metropolis-Hastings** the system is trained with contrastive divergence and allowed to wander *one* step from the ground-truth configuration. Once the parameters are learned, MAP inference is performed using Metropolis-Hastings (with a proposal distribution that modifies a single variable at a time).

- **SampleRank with Metropolis-Hastings (MH-SR):** this system is the same as above, but trains the CRF using SampleRank rather than CD1. MAP is performed with Metropolis-Hastings using a proposal distribution that modifies a single variable at a time (same proposer as in MH-CD1).

- **Reinforcement Learning (RL):** this is the system introduced in the paper that trains the CRF with delayed reward using $Q(\lambda)$ to learn state-action returns. The actions are derived from the same proposal distribution as used by our Metropolis-Hastings (MH-CD1,MH-SR) systems (modifying a single variable at a time); however it is exhaustively applied to find the maximum action. We set the RL parameters as follows: $\alpha$=0.00001, $\lambda$=0.9, $\gamma$=0.9.

- **GLUE:** in order to compare with a well-known system on the this dataset, we choose GLUE [24].

In these experiments contrastive divergence and SampleRank were run for 10,000 samples each , while reinforcement learning was run for twenty episodes and 200 steps per episode. CD1 and SampleRank were run for more steps to compensate for only observing a single action at each step (recall RL computes the action with the maximum value at each step by observing a large number of samples).

## 4.3 Results

In Table 1 we compare F1 (pairwise-matching) scores of the various systems on the course catalog and company profile datasets. We also compare to the well known system, GLUE [24]. SampleRank (MH-SR), contrastive divergence (MH-CD1) and reinforcement learning (RL) underwent ten training episodes initialized from random configurations; during MAP inference we initialized the systems to the state predicted by greedy agglomerative clustering. Both SampleRank and reinforcement learning were able to achieve higher scores than greedy; however, reinforcement learning outperformed all systems with an error reduction of 75.3% over contrastive divergence, 28% over SampleRank, 71% over GLUE and 48% over the previous state of the art (greedy agglomerative inference on a conditional random field). Reinforcement learning also reduces error over each system on the company profile dataset.

After observing the improvements obtained by reinforcement learning, we wished to test how robust the method was at recovering from the local optima problem described in the introduction. To gain more insight, we designed a separate experiment to compare Metropolis-Hastings inference (trained with SampleRank) and reinforcement learning more carefully.

In the second experiment we evaluate our approach under more difficult conditions. In particular, the MAP inference procedures are initialized to random clusterings (in regions riddled with the type of local optima discussed in the introduction). We then compare greedy MAP inference on a model whose parameters were learned with RL, to Metropolis-Hastings on a model with parameters learned from SampleRank. More specifically, we generate a set of ten random configurations from the test corpus and run both algorithms, averaging the results over the ten runs. The first two rows of Table 2 summarizes this experiment. Even though reinforcement learning's policy requires it to be greedy with respect to the q-function, we observe that it is able to better escape the random initial configuration than the Metropolis-Hastings method. This is demonstrated in the first rows of Table 2. Although both systems perform worse than under these conditions than those of the previous experiment, reinforcement learning does much better in this situation, indicating that the q-function learned is fairly robust and capable of generalizing to random regions of the space.

After observing Metropolis-Hasting's tendency to get stuck in regions of lower score than reinforcement learning, we test RL to see if it would fall victim to these same optima. In the last two rows of Table 2 we record the results of re-running both reinforcement learning and Metropolis-Hastings (on the SampleRank model) from the configurations Metropolis-Hastings became stuck. We notice that RL is able to climb out of these optima and achieve a score comparable to our first experiment.

MH is also able to progress out of the optima, demonstrating that the stochastic method is capable of escaping optima, but perhaps not as quickly on this particular problem.

|  | F1 | Precision | Recall |
|---|---|---|---|
| RL on random | **86.4** | 87.2 | 85.6 |
| MH-SR on random | 81.1 | 82.9 | 79.3 |
| RL on MH-SR | **93.0** | 94.6 | 91.5 |
| MH-SR on MH-SR | 84.3 | 87.3 | 81.5 |

Table 2: Average pairwise-matching precision, recall and F1 over ten random initialization points, and on the output of MH-SR after 10,000 inference steps.

# 5   Related Work

The expanded version of this work is our technical report [17], which provides additional detail and motivation. Our approach is similar in spirit to Zhang and Dietterich who propose a reinforcement learning framework for solving combinatorial optimization problems [25]. Similar to this approach, we also rely on generalization techniques in RL in order to directly approximate a policy over unseen test domains. However, our formulation provides a framework that explicitly targets the MAP problem in large factor graphs and takes advantage of the log-linear representation of such models in order to employ a well studied class of generalization techniques in RL known as linear function approximation. Learning generalizable function approximators has been also studied for efficiently guiding standard search algorithms through experience [26].

There are a number of approaches for learning parameters that specifically target the problem of MAP inference. For example, the frameworks of *LASO* [27] and *SEARN* [28]) formulate MAP in the context of search optimization, where a cost function is learned to score partial (incomplete) configurations that lead to a goal state. In this framework, actions incrementally construct a solution, rather than explore the solution space itself. As shown in [28] these frameworks have connections to learning policies in reinforcement learning. However, the policies are learned over incomplete configurations. In contrast, we formulate parameter learning in factor graphs as an MDP over the space of *complete* configurations from which a variety of RL methods can be used to set the parameters.

Another approach that targets the problem of MAP inference is SampleRank [11, 23], which computes atomic gradient updates from jumps in the local search space. This method has the advantage of learning over the space of complete configurations, but ignores the issue of delayed reward.

# 6   Conclusions and Future Work

We proposed an approach for solving the MAP inference problem in large factor graphs by using reinforcement learning to train model parameters. RL allows us to evaluate jumps in the configuration space based on a value function that optimizes the long term improvement in model scores. Hence – unlike most search optimization approaches – the system is able to move out of local optima while aiming for the MAP configuration. Benefitting from log linear nature of factor graphs such as CRFs we are also able to employ well studied RL linear function approximation techniques for learning generalizable value functions that are able to provide value estimates on the test set. Our experiments over a real world domain shows impressive error reduction when compared to the other approaches. Future work should investigate additional RL paradigms for training models such as actor-critic.

## Acknowledgments

This work was supported in part by the CIIR; SRI #27-001338 and ARFL #FA8750-09-C-0181, CIA, NSA and NSF #IIS-0326249; Army #W911NF-07-1-0216 and UPenn subaward #103-548106; and UPenn NSF #IS-0803847. Any opinions, findings and conclusions or recommendations expressed in this material are the authors' and do not necessarily reflect those of the sponsor.

## Footnotes

[1]http://pages.cs.wisc.edu/ anhai/wisc-si-archive/

# References

[1] Andrew McCallum, Dayne Freitag, and Fernando Pereira. Maximum entropy markov models for information extraction and segmentation. In *International Conference on Machine Learning (ICML)*, 2000.

[2] John D. Lafferty, Andrew McCallum, and Fernando Pereira. Conditional random fields: Probabilistic models for segmenting and labeling sequence data. In *Int Conf on Machine Learning (ICML)*, 2001.

[3] Ben Taskar, Carlos Guestrin, and Daphne Koller. Max-margin markov networks. In *NIPS*, 2003.

[4] Ryan McDonald and Fernando Pereira. Online learning of approximate dependency parsing algorithms. In *European Chapter of the Association for Computational Linguistics (EACL)*, pages 81–88, 2006.

[5] Matthew Richardson and Pedro Domingos. Markov logic networks. *Machine Learning*, 62, 2006.

[6] Brian Milch, Bhaskara Marthi, and Stuart Russell. *BLOG: Relational Modeling with Unknown Objects*. PhD thesis, University of California, Berkeley, 2006.

[7] Andrew McCallum, Khashayar Rohanimanesh, Michael Wick, Karl Schultz, and Sameer Singh. Factorie: Efficient probabilistic programming via imperative declarations of structure, inference and learning. In *Neural Information Processing Systems(NIPS) Workshop on Probabilistic Programming*, Vancouver, BC, Canda, 2008.

[8] Aria Haghighi and Dan Klein. Unsupervised coreference resolution in a nonparametric bayesian model. In *Association for Computational Linguistics (ACL)*, 2007.

[9] Hanna Pasula, Bhaskara Marthi, Brian Milch, Stuart Russell, and Ilya Shpitser. Identity uncertainty and citation matching. In *Advances in Neural Information Processing Systems 15*. MIT Press, 2003.

[10] Sonia Jain and Radford M. Neal. A split-merge markov chain monte carlo procedure for the dirichlet process mixture model. *Journal of Computational and Graphical Statistics*, 13:158–182, 2004.

[11] Aron Culotta. *Learning and inference in weighted logic with application to natural language processing*. PhD thesis, University of Massachusetts, May 2008.

[12] Richard S. Sutton and Andrew G. Barto. *Reinforcement Learning: An Introduction*. The MIT Press, March 1998.

[13] Richard S. Sutton. Learning to predict by the methods of temporal differences. *Machine Learning*, pages 9–44, 1988.

[14] Robert H. Crites and Andrew G. Barto. Improving elevator performance using reinforcement learning. In *Advances in Neural Information Processing Systems 8*, pages 1017–1023. MIT Press, 1996.

[15] Wei Zhang and Thomas G. Dietterich. Solving combinatorial optimization tasks by reinforcement learning: A general methodology applied to resource-constrained scheduling. *Journal of Artificial Intelligence Reseach*, 1, 2000.

[16] Gerald Tesauro. Temporal difference learning and td-gammon. *Commun. ACM*, 38(3):58–68, 1995.

[17] Khashayar Rohanimanesh, Michael Wick, Sameer Singh, and Andrew McCallum. Reinforcement learning for map inference in large factor graphs. Technical Report #UM-CS-2008-040, University of Massachusetts, Amherst, 2008.

[18] Christopher J. Watkins. *Learning from Delayed Rewards*. PhD thesis, Kings College, Cambridge, 1989.

[19] Christopher J. Watkins and Peter Dayan. Q-learning. *Machine Learning*, 8(3):279–292, May 1992.

[20] Andrew McCallum and Charles Sutton. Piecewise training with parameter independence diagrams: Comparing globally- and locally-trained linear-chain CRFs. In *NIPS Workshop on Learning with Structured Outputs*, 2004.

[21] Geoffrey E. Hinton. Training products of experts by minimizing contrastive divergence. *Neural Computation*, 14(8):1771–1800, 2002.

[22] Culotta. First. In *International Joint Conference on Artificial Intelligence*, 2007.

[23] Khashayar Rohanimanesh, Michael Wick, and Andrew McCallum. Inference and learning in large factor graphs with adaptive proposal distributions. Technical Report #UM-CS-2009-028, University of Massachusetts, Amherst, 2009.

[24] AnHai Doan, Jayant Madhavan, Pedro Domingos, and Alon Y. Halevy. Learning to map between ontologies on the semantic web. In *WWW*, page 662, 2002.

[25] Wei Zhang and Thomas G. Dietterich. A reinforcement learning approach to job-shop scheduling. In *International Joint Conference on Artificial Intelligence (IJCAI)*, pages 1114–1120, 1995.

[26] Justin Boyan and Andrew W. Moore. Learning evaluation functions to improve optimization by local search. *J. Mach. Learn. Res.*, 1:77–112, 2001.

[27] Hal Daumé III and Daniel Marcu. Learning as search optimization: approximate large margin methods for structured prediction. In *International Conference on Machine learning (ICML)*, 2005.

[28] Hal Daumé III, John Langford, and Daniel Marcu. Search-based structured prediction. *Machine Learning*, 2009.

